# LEARNING A COLOR ALGORITHM FROM EXAMPLES

## Anya C. Hurlbert and Tomaso A. Poggio
Artificial Intelligence Laboratory and Department of Brain and Cognitive Sciences,
Massachusetts Institute of Technology, Cambridge, Massachusetts 02139, USA

## ABSTRACT

A lightness algorithm that separates surface reflectance from illumination in a Mondrian world is synthesized automatically from a set of examples, pairs of input (image irradiance) and desired output (surface reflectance). The algorithm, which resembles a new lightness algorithm recently proposed by Land, is approximately equivalent to filtering the image through a center-surround receptive field in individual chromatic channels. The synthesizing technique, optimal linear estimation, requires only one assumption, that the operator that transforms input into output is linear. This assumption is true for a certain class of early vision algorithms that may therefore be synthesized in a similar way from examples. Other methods of synthesizing algorithms from examples, or "learning", such as backpropagation, do not yield a significantly different or better lightness algorithm in the Mondrian world. The linear estimation and backpropagation techniques both produce simultaneous brightness contrast effects.

The problems that a visual system must solve in decoding two-dimensional images into three-dimensional scenes (inverse optics problems) are difficult: the information supplied by an image is not sufficient by itself to specify a unique scene. To reduce the number of possible interpretations of images, visual systems, whether artificial or biological, must make use of natural constraints, assumptions about the physical properties of surfaces and lights. Computational vision scientists have derived effective solutions for some inverse optics problems (such as computing depth from binocular disparity) by determining the appropriate natural constraints and embedding them in algorithms. How might a visual system discover and exploit natural constraints on its own? We address a simpler question: Given only a set of examples of input images and desired output solutions, can a visual system synthesize, or "learn", the algorithm that converts input to output? We find that an algorithm for computing color in a restricted world can be constructed from examples using standard techniques of optimal linear estimation.

The computation of color is a prime example of the difficult problems of inverse optics. We do not merely discriminate between different wavelengths of light; we assign

roughly constant colors to objects even though the light signals they send to our eyes change as the illumination varies across space and chromatic spectrum. The computational goal underlying color constancy seems to be to extract the invariant surface spectral reflectance properties from the image irradiance, in which reflectance and illumination are mixed[1].

Lightness algorithms [2-8], pioneered by Land, assume that the color of an object can be specified by its lightness, or relative surface reflectance, in each of three independent chromatic channels, and that lightness is computed in the same way in each channel. Computing color is thereby reduced to extracting surface reflectance from the image irradiance in a single chromatic channel.

The image irradiance, $s'$, is proportional to the product of the illumination intensity $e'$ and the surface reflectance $r'$ in that channel:

$$s'(x,y) = r'(x,y)e'(x,y). \tag{1}$$

This form of the image intensity equation is true for a Lambertian reflectance model, in which the irradiance $s'$ has no specular components, and for appropriately chosen color channels [9]. Taking the logarithm of both sides converts it to a sum:

$$s(x,y) = r(x,y) + e(x,y), \tag{2}$$

where $s = log(s')$, $r = log(r')$ and $e = log(e')$.

Given $s(x,y)$ alone, the problem of solving Eq. 2 for $r(x,y)$ is underconstrained. Lightness algorithms constrain the problem by restricting their domain to a world of Mondrians, two-dimensional surfaces covered with patches of random colors[2] and by exploiting two constraints in that world: (i) $r'(x,y)$ is uniform within patches but has sharp discontinuities at edges between patches and (ii) $e'(x,y)$ varies smoothly across the Mondrian. Under these constraints, lightness algorithms can recover a good approximation to $r(x,y)$ and so can recover lightness triplets that label roughly constant colors [10].

We ask whether it is possible to synthesize from examples an algorithm that extracts reflectance from image irradiance, and whether the synthesized algorithm will resemble existing lightness algorithms derived from an explicit analysis of the constraints. We make one assumption, that the operator that transforms irradiance into reflectance is linear. Under that assumption, motivated by considerations discussed later, we use optimal linear estimation techniques to synthesize an operator from examples. The examples are pairs of images: an input image of a Mondrian under illumination that varies smoothly across space and its desired output image that displays the reflectance of the Mondrian without the illumination. The technique finds the linear estimator that best maps input into desired output, in the least squares sense.

For computational convenience we use one-dimensional "training vectors" that represent vertical scan lines across the Mondrian images (Fig. 1). We generate many

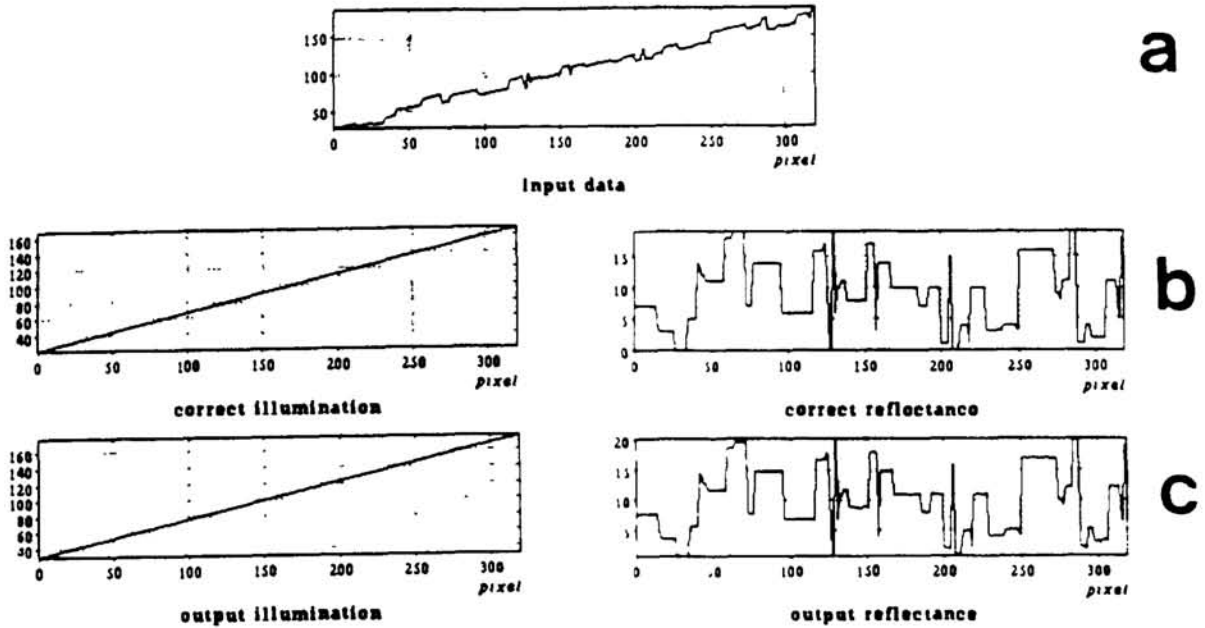

**Input data**

**correct illumination**

**correct reflectance**

**output illumination**

**output reflectance**

**a**

**b**

**c**

Fig. 1. (a) The input data, a one-dimensional vector 320 pixels long. Its random Mondrian reflectance pattern is superimposed on a linear illumination gradient with a random slope and offset. (b) shows the corresponding output solution, on the left the illumination and on the right reflectance. We used 1500 such pairs of input-output examples (each different from the others) to train the operator shown in Fig. 2. (c) shows the result obtained by the estimated operator when it acts on the input data (a), not part of the training set. On the left is the illumination and on the right the reflectance, to be compared with (b). This result is fairly typical: in some cases the prediction is even better, in others it is worse.

different input vectors $s$ by adding together different random $r$ and $e$ vectors, according to Eq. 2. Each vector $r$ represents a pattern of step changes across space, corresponding to one column of a reflectance image. The step changes occur at random pixels and are of random amplitude between set minimum and maximum values. Each vector $e$ represents a smooth gradient across space with a random offset and slope, corresponding to one column of an illumination image. We then arrange the training vectors $s$ and $r$ as the columns of two matrices $S$ and $R$, respectively. Our goal is then to compute the optimal solution $L$ of

$$LS = R \qquad (3)$$

where $L$ is a linear operator represented as a matrix.

It is well known that the solution of this equation that is optimal in the least squares sense is

$$L = RS^+ \tag{4}$$

where $S^+$ is the Moore-Penrose pseudoinverse [11]. We compute the pseudoinverse by overconstraining the problem – using many more training vectors than there are number of pixels in each vector – and using the straightforward formula that applies in the overconstrained case [12]: $S^+ = S^T(SS^T)^{-1}$.

The operator $L$ computed in this way recovers a good approximation to the correct output vector $r$ when given a new $s$, not part of the training set, as input (Fig. 1c). A second operator, estimated in the same way, recovers the illumination $e$. Acting on a random two-dimensional Mondrian $L$ also yields a satisfactory approximation to the correct output image.

Our estimation scheme successfully synthesizes an algorithm that performs the lightness computation in a Mondrian world. *What* is the algorithm and what is its relationship to other lightness algorithms? To answer these questions we examine the structure of the matrix $L$. We assume that, although the operator is *not* a convolution operator, it should approximate one far from the boundaries of the image. That is, in its central part, the operator should be space-invariant, performing the same action on each point in the image. Each row in the central part of $L$ should therefore be the same as the row above but displaced by one element to the right. Inspection of the matrix confirmes this expectation. To find the form of $L$ in its center, we thus average the rows there, first shifting them appropriately. The result, shown in Fig. 2, is a space-invariant filter with a narrow positive peak and a broad, shallow, negative surround.

Interestingly, the filter our scheme synthesizes is very similar to Land's most recent retinex operator [5], which divides the image irradiance at each pixel by a weighted average of the irradiance at all pixels in a large surround and takes the logarithm of that result to yield lightness [13]. The lightness triplets computed by the retinex operator agree well with human perception in a Mondrian world. The retinex operator and our matrix $L$ both differ from Land's earlier retinex algorithms, which require a non-linear thresholding step to eliminate smooth gradients of illumination.

The shape of the filter in Fig. 2, particularly of its large surround, is also suggestive of the "nonclassical" receptive fields that have been found in V4, a cortical area implicated in mechanisms underlying color constancy [14-17].

The form of the space-invariant filter is similar to that derived in our earlier formal analysis of the lightness problem [8]. It is qualitatively the same as that which results from the direct application of regularization methods exploiting the spatial constraints on reflectance and illumination described above [9,18,19]. The Fourier transform of the filter of Fig. 2 is approximately a bandpass filter that cuts out low frequencies due

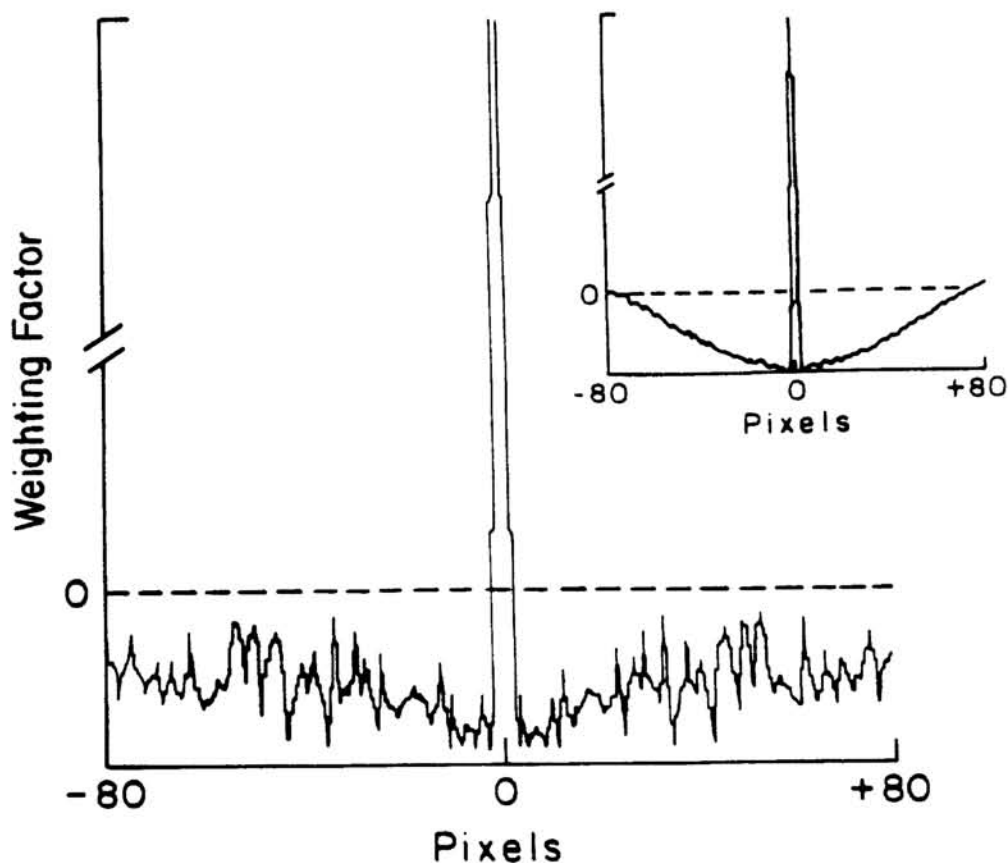

Fig. 2. The space-invariant part of the estimated operator, obtained by shifting and averaging the rows of a 160-pixel-wide central square of the matrix $L$, trained on a set of 1500 examples with linear illumination gradients (see Fig. 1). When logarithmic illumination gradients are used, a qualitatively similar receptive field is obtained. In a separate experiment we use a training set of one-dimensional Mondrians with either linear illumination gradients or slowly varying sinusoidal illumination components with random wavelength, phase and amplitude. The resulting filter is shown in the inset. The surrounds of both filters extend beyond the range we can estimate reliably, the range we show here.

to slow gradients of illumination and preserves intermediate frequencies due to step changes in reflectance. In contrast, the operator that recovers the illumination, $e$, takes the form of a low-pass filter. We stress that the entire operator $L$ is not a space-invariant filter.

In this context, it is clear that the shape of the estimated operator should vary with the type of illumination gradient in the training set. We synthesize a second operator using a new set of examples that contain equal numbers of vectors with random, sinusoidally varying illumination components and vectors with random, linear illumination gradients. Whereas the first operator, synthesized from examples with strictly linear illumination gradients, has a broad negative surround that remains virtually constant throughout its extent, the new operator's surround (Fig. 2, inset) has a smaller extent

and decays smoothly towards zero from its peak negative value in its center.

We also apply the operator in Fig. 2 to new input vectors in which the density and amplitude of the step changes of reflectance differ greatly from those on which the operator is trained. The operator performs well, for example, on an input vector representing one column of an image of a small patch of one reflectance against a uniform background of a different reflectance, the entire image under a linear illumination gradient. This result is consistent with psychophysical experiments that show that color constancy of a patch holds when its Mondrian background is replaced by an equivalent grey background [20].

The operator also produces simultaneous brightness contrast, as expected from the shape and sign of its surround. The output reflectance it computes for a patch of fixed input reflectance decreases linearly with increasing average irradiance of the input test vector in which the patch appears. Similarly, to us, a dark patch appears darker when against a light background than against a dark one.

This result takes one step towards explaining such illusions as the Koffka Ring [21]. A uniform gray annulus against a bipartite background (Fig. 3a) appears to split into two halves of different lightnesses when the midline between the light and dark halves of the background is drawn across the annulus (Fig. 3b). The estimated operator acting on the Koffka Ring of Fig. 3b reproduces our perception by assigning a lower output reflectance to the left half of the annulus (which appears darker to us) than to the right half [22]. Yet the operator gives this brightness contrast effect whether or not the midline is drawn across the annulus (Fig. 3c). Because the operator can perform only a linear transformation between the input and output images, it is not surprising that the addition of the midline in the input evokes so little change in the output. These results demonstrate that the linear operator alone cannot compute lightness in all worlds and suggest that an additional operator might be necessary to mark and guide it within bounded regions.

Our estimation procedure is motivated by our previous observation [9,23,18] that standard regularization algorithms [19] in early vision define linear mappings between input and output and therefore can be estimated associatively under certain conditions. The technique of optimal linear estimation that we use is closely related to optimal Bayesian estimation [9]. If we were to assume from the start that the optimal linear operator is space-invariant, we could considerably simplify (and streamline) the computation by using standard correlation techniques [9,24].

How does our estimation technique compare with other methods of "learning" a lightness algorithm? We can compute the regularized pseudoinverse using gradient descent on a "neural" network [25] with linear units. Since the pseudoinverse is the unique best linear approximation in the $L_2$ norm, a gradient descent method that

minimizes the square error between the actual output and desired output of a fully connected linear network is guaranteed to converge, albeit slowly. Thus gradient descent in weight space converges to the same result as our first technique, the global minimum.

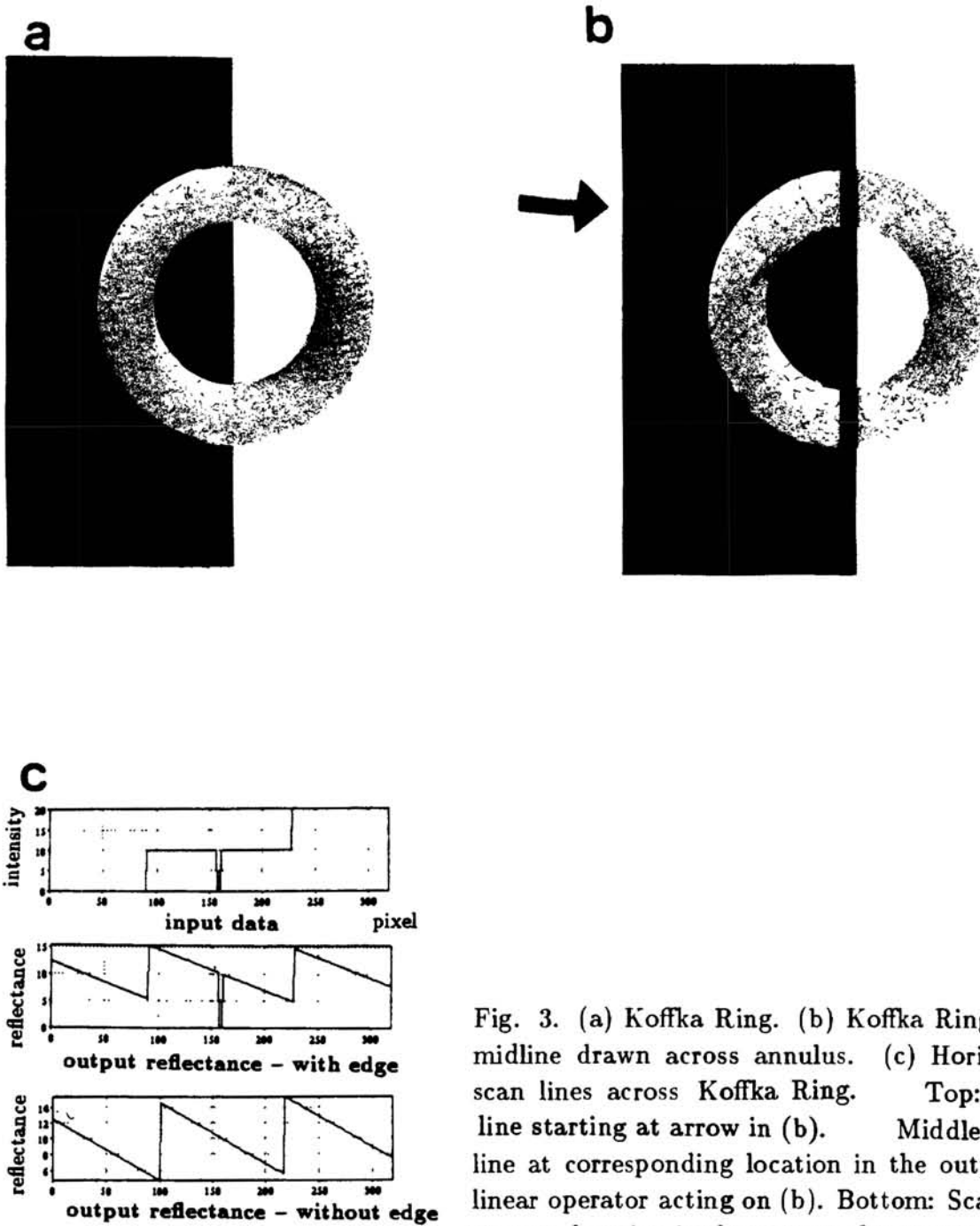

Fig. 3. (a) Koffka Ring. (b) Koffka Ring with midline drawn across annulus. (c) Horizontal scan lines across Koffka Ring. Top: Scan line starting at arrow in (b). Middle: Scan line at corresponding location in the output of linear operator acting on (b). Bottom: Scan line at same location in the output of operator acting on (a).

We also compare the linear estimation technique with a "backpropagation" network: gradient descent on a 2-layer network with sigmoid units [25] (32 inputs, 32 "hidden units", and 32 linear outputs), using training vectors 32 pixels long. The network requires an order of magnitude more time to converge to a stable configuration than does the linear estimator for the same set of 32-pixel examples. The network's performance is slightly, yet consistently, better, measured as the root-mean-square error in output, averaged over sets of at least 2000 new input vectors. Interestingly, the backpropagation network and the linear estimator err in the same way on the same input vectors. It is possible that the backpropagation network may show considerable inprovement over the linear estimator in a world more complex than the Mondrian one. We are presently examining its performance on images with real-world features such as shading, shadows, and highlights[26].

We do not think that our results mean that color constancy may be learned during a critical period by biological organisms. It seems more reasonable to consider them simply as a demonstration on a toy world that in the course of evolution a visual system may recover and exploit natural constraints hidden in the physics of the world. The significance of our results lies in the facts that a simple statistical technique may be used to synthesize a lightness algorithm from examples; that the technique does as well as other techniques such as backpropagation; and that a similar technique may be used for other problems in early vision. Furthermore, the synthesized operator resembles both Land's psychophysically-tested retinex operator and a neuronal nonclassical receptive field. The operator's properties suggest that simultaneous color (or brightness) contrast might be the result of the visual system's attempt to discount illumination gradients [27].

## REFERENCES AND NOTES

1. Since we do not have perfect color constancy, our visual system must not extract reflectance exactly. The limits on color constancy might reveal limits on the underlying computation.

2. E.H. Land, *Am. Sci.* **52**, 247 (1964).

3. E.H. Land and J.J. McCann, *J. Opt. Soc. Am.* **61**, 1 (1971).

4. E.H. Land, in *Central and Peripheral Mechanisms of Colour Vision*, T. Ottoson and S. Zeki, Eds., (Macmillan, New York, 1985), pp. 5-17.

5. E.H. Land, *Proc. Nat. Acad. Sci. USA* **83**, 3078 (1986).

6. B.K.P. Horn, *Computer Graphics and Image Processing* **3**, 277 (1974).

7. A. Blake, in *Central and Peripheral Mechanisms of Colour Vision*, T. Ottoson and S. Zeki, Eds., (Macmillan, New York, 1985), pp. 45-59.

8. A. Hurlbert, *J. Opt. Soc. Am. A* **3**, 1684 (1986).

9. A. Hurlbert and T. Poggio, *Artificial Intelligence Laboratory Memo 909*, (M.I.T., Cambridge, MA, 1987).

10. $r'(x,y)$ can be recovered at best only to within a constant, since Eq. 1 is invariant under the transformation of $r'$ into $ar'$ and $e'$ into $a^{-1}e'$, where $a$ is a constant.

11. A. Albert, *Regression and the Moore-Penrose Pseudoinverse*, (Academic Press, New York, 1972).

12. The pseudoinverse, and therefore L, may also be computed by recursive techniques that improve its form as more data become available[11].

13. Our synthesized filter is not exactly identical with Land's: the filter of Fig. 2 subtracts from the value at each point the average value of the logarithm of irradiance at all pixels, rather than the logarithm of the average values. The estimated operator is therefore linear in the logarithms, whereas Land's is not. The numerical difference between the outputs of the two filters is small in most cases (Land, personal communication), and both agree well with psychophysical results.

14. R. Desimone, S.J. Schein, J. Moran and L.G. Ungerleider, *Vision Res.* **25**, 441 (1985).

15. H.M. Wild, S.R. Butler, D. Carden and J.J. Kulikowski, *Nature (London)* **313**, 133 (1985).

16. S.M. Zeki, *Neuroscience* **9**, 741 (1983).

17. S.M. Zeki, *Neuroscience* **9**, 767 (1983).

18. T. Poggio, et. al., in *Proceedings Image Understanding Workshop*, L. Baumann, Ed., (Science Applications International Corporation, McLean, VA, 1985), pp. 25-39.

19. T. Poggio, V. Torre and C. Koch, *Nature (London)* **317**, 314 (1985).

20. A. Valberg and B. Lange-Malecki, *Investigative Ophthalmology and Visual Science Supplement* **28**, 92 (1987).

21. K. Koffka, *Principles of Gestalt Psychology*, (Harcourt, Brace and Co., New York, 1935).

22. Note that the operator achieves this effect by subtracting a non-existent illumination gradient from the input signal.

23. T. Poggio and A. Hurlbert, *Artificial Intelligence Laboratory Working Paper 264*, (M.I.T., Cambridge, MA, 1984).

24. Estimation of the operator on two-dimensional examples is possible, but computationally very expensive if done in the same way. The present computer simulations require several hours when run on standard serial computers. The two-dimensional case

will need much more time (our one-dimensional estimation scheme runs orders of magnitude faster on a CM-1 Connection Machine System with 16K-processors).

25. D. E. Rumelhart, G.E. Hinton and R.J. Williams, *Nature (London)* **323**, 533 (1986).

26. A. Hurlbert, *The Computation of Color*, Ph.D. Thesis, M.I.T., Cambridge, MA, in preparation.

27. We are grateful to E. Land, E. Hildreth. J. Little, F. Wilczek and D. Hillis for reading the draft and for useful discussions. A. Rottenberg developed the routines for matrix operations that we used on the Connection Machine. T. Breuel wrote the backpropagation simulator.